# Factorial Learning by Clustering Features

**Joshua B. Tenenbaum and Emanuel V. Todorov**
Department of Brain and Cognitive Sciences
Massachusetts Institute of Technology
Cambridge, MA 02139
{jbt,emo}@psyche.mit.edu

## Abstract

We introduce a novel algorithm for factorial learning, motivated by segmentation problems in computational vision, in which the underlying factors correspond to clusters of highly correlated input features. The algorithm derives from a new kind of competitive clustering model, in which the cluster generators compete to explain each feature of the data set and cooperate to explain each input example, rather than competing for examples and cooperating on features, as in traditional clustering algorithms. A natural extension of the algorithm recovers hierarchical models of data generated from multiple unknown categories, each with a different, multiple causal structure. Several simulations demonstrate the power of this approach.

## 1  INTRODUCTION

Unsupervised learning is the search for structure in data. Most unsupervised learning systems can be viewed as trying to invert a particular generative model of the data in order to recover the underlying causal structure of their world. Different learning algorithms are then primarily distinguished by the different generative models they embody, that is, the different kinds of structure they look for.

*Factorial learning*, the subject of this paper, tries to find a set of independent causes that cooperate to produce the input examples. We focus on *strong* factorial learning, where the goal is to recover the actual degrees of freedom responsible for generating the observed data, as opposed to the more general *weak* approach, where the goal

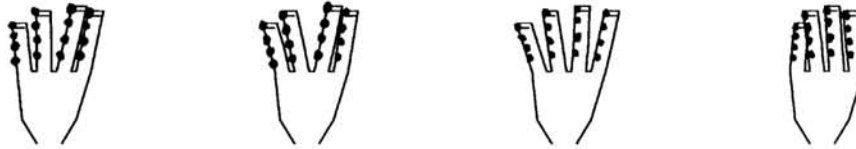

Figure 1: A simple factorial learning problem. The learner observes an articulated hand in various configurations, with each example specified by the positions of 16 tracked features (shown as black dots). The learner might recover four underlying factors, corresponding to the positions of the fingers, each of which claims responsiblity for four features of the data set.

is merely to recover some factorial model that explains the data efficiently. Strong factorial learning makes a claim about the nature of the world, while weak factorial learning only makes a claim about the nature of the learner's representations (although the two are clearly related). Standard subspace algorithms, such as principal component analysis, fit a linear, factorial model to the input data, but can only recover the true causal structure in very limited situations, such as when the data are generated by a linear combination of independent factors with significantly different variances (as in signal-from-noise separation).

Recent work in factorial learning suggests that the general problem of recovering the true, multiple causal structure of an arbitrary, real-world data set is very difficult, and that specific approaches must be tailored to specific, but hopefully common, classes of problems (Foldiak, 1990; Saund, 1995; Dayan and Zemel, 1995). Our own interest in multiple cause learning was motivated by segmentation problems in computational vision, in which the underlying factors correspond ideally to disjoint clusters of highly correlated input features. Examples include the segmentation of articulated objects into functionally independent parts, or the segmentation of multiple-object motion sequences into tracks of individual objects. These problems, as well as many other problems of pattern recognition and analysis, share a common set of constraints which makes factorial learning both appropriate and tractable. Specifically, while each observed example depends on some combination of several factors, any one input feature always depends on only one such factor (see Figure 1). Then the generative model decomposes into independent sets of functionally grouped input features, or *functional parts* (Tenenbaum, 1994).

In this paper, we propose a learning algorithm that extracts these functional parts. The key simplifying assumption, which we call the *membership constraint*, states that each feature belongs to at most one functional part, and that this membership is constant over the set of training examples. The membership constraint allows us to treat the factorial learning problem as a novel kind of clustering problem. The cluster generators now compete to explain each feature of the data set and cooperate to explain each input example, rather than competing for examples and cooperating on features, as in traditional clustering systems such as $K$-means or mixture models. The following sections discuss the details of the feature clustering algorithm for extracting functional parts, a simple but illustrative example, and extensions. In particular, we demonstrate a natural way to relax the strict membership constraint and thus learn hierarchical models of data generated from multiple unknown categories, each with a different multiple causal structure.

## 2    THE FEATURE CLUSTERING ALGORITHM

Our algorithm for extracting functional parts derives from a statistical mechanics formulation of the soft clustering problem (inspired by Rose, Gurewitz, and Fox, 1990; Hinton and Zemel, 1994). We take as input a data set $\{x_j^{(i)}\}$, with $I$ examples of $J$ real-valued features. The best $K$-cluster representation of these $J$ features is given by an optimal set of cluster parameters, $\{\theta_k\}$, and an optimal set of assignments, $\{p_{jk}\}$. The assignment $p_{jk}$ specifies the probability of assigning feature $j$ to cluster $k$, and depends directly on $E_{jk} = \sum_i (x_j^{(i)} - f_{jk}^{(i)})^2$, the total squared difference (over the $I$ training examples) between the observed feature values $x_j^{(i)}$ and cluster $k$'s predictions $f_{jk}^{(i)}$. The parameters $\theta_k$ define cluster $k$'s generative model, and thus determine the predictions $f_{jk}^{(i)}(\theta_k)$.

If we limit functional parts to clusters of *linearly* correlated features, then the appropriate generative model has $f_{jk}^{(i)} = w_{jk} y_k^{(i)} + u_j$, with cluster parameters $\theta_k = \{y_k^{(i)}, w_{jk}, u_j\}$ to be estimated. That is, for each example $i$, part $k$ predicts the value of input feature $j$ as a linear function of some part-specific factor $y_k^{(i)}$ (such as finger position in Figure 1). For the purposes of this paper, we assume zero-mean features and ignore the $u_j$ terms. Then $E_{jk} = \sum_i (x_j^{(i)} - w_{jk} y_k^{(i)})^2$.

The optimal cluster parameters and assignments can now be found by maximizing the complete log likelihood of the data given the $K$-cluster representation, or equivalently, in the framework of statistical mechanics, by minimizing the free energy

$$F = E - \frac{1}{\beta}H = \sum_j \sum_k p_{jk}(E_{jk} + \frac{1}{\beta}\log p_{jk}) \qquad (1)$$

subject to the membership constraints, $\sum_k p_{jk} = 1, (\forall j)$. Minimizing the energy,

$$E = \sum_j \sum_k p_{jk} E_{jk}, \qquad (2)$$

reduces the expected reconstruction error, leading to more accurate representations. Maximizing the entropy,

$$H = -\sum_j \sum_k p_{jk} \log p_{jk}, \qquad (3)$$

distributes responsibility for each feature across many parts, thus decreasing the independence of the parts and leading to simpler representations (with fewer degrees of freedom). In line with Occam's Razor, minimizing the energy-entropy tradeoff finds the representation that, at a particular temperature $1/\beta$, best satisfies the conflicting requirements of low error and low complexity.

We minimize the free energy with a generalized EM procedure (Neal and Hinton, 1994), setting derivatives to zero and iterating the resulting update equations:

$$p_{jk} = \frac{e^{-\beta E_{jk}}}{\sum_k e^{-\beta E_{jk}}} \qquad (4)$$

$$y_k^{(i)} = \sum_j p_{jk} w_{jk} x_j^{(i)} \qquad (5)$$

$$w_{jk} = \sum_i x_j^{(i)} y_k^{(i)} . \qquad (6)$$

This update procedure assumes a normalization step $y_k^{(i)} = y_k^{(i)}/(\sum_{i'}(y_k^{(i')})^2)^{1/2}$ in each iteration, because without some additional constraint on the magnitudes of $y_k^{(i)}$ (or $w_{jk}$), inverting the generative model $f_{jk}^{(i)} = w_{jk}y_k^{(i)}$ is an ill-posed problem.

This algorithm maps naturally onto a simple network architecture. The hidden unit activities, representing the part-specific factors $y_k^{(i)}$, are computed from the observations $x_j^{(i)}$ via bottom-up weights $p_{jk}w_{jk}$, normalized, and multiplied by top-down weights $w_{jk}$ to generate the network's predictions $f_{jk}^{(i)}$. The weights adapt according to a hybrid learning rule, with $w_{jk}$ determined by a Hebb rule (as in subspace learning algorithms), and $p_{jk}$ determined by a competitive, softmax function of the reconstruction error $E_{jk}$ (as in soft mixture models).

# 3   LEARNING A HIERARCHY OF PARTS

The following simulation illustrates the algorithm's behavior on a simple, part segmentation task. The training data consist of 60 examples with 16 features each, representing the horizontal positions of 16 points on an articulated hand in various configurations (as in Figure 1). The data for this example were generated by a hierarchical, random process that produced a low correlation between all 16 features, a moderate correlation between the four features on each finger, and a high correlation between the two features on each joint (two joints per finger). To fully explain this data set, the algorithm should be able to find a corresponding hierarchy of increasingly complex functional part representations.

To evaluate the network's representation of this data set, we inspect the learned weights $p_{jk}w_{jk}$, which give the total contribution of feature $j$ to part $k$ in (5). In Figure 2, these weights are plotted for several different values of $\beta$, with gray boxes indicating zero weights, white indicating strong positive weights, and black indicating strong negative weights. The network was configured with $K = 16$ part units, to ensure that all potential parts could be found. When fewer than $K$ distinct parts are found, some of the cluster units have identical parameters (appearing as identical columns in Figure 2). These results were generated by deterministic annealing, starting with $\beta \ll 1$, and perturbing the weights slightly each time $\beta$ was increased, in order to break symmetries.

Figure 2 shows that the number of distinct parts found increases with $\beta$, as more accurate (and more complex) representations become favored. In (4), we see that $\beta$ controls the number of distinct parts via the strength of the competition for features. At $\beta = 0$, every part takes equal responsibility for every feature. Without competition, there can be no diversity, and thus only one distinct part is discovered at low $\beta$, corresponding to the whole hand (Figure 2a). As $\beta$ increases, the competition for features gets stiffer, and parts split into their component subparts. The network finds first four distinct parts (with four features each), corresponding to individual fingers (Figure 2c), and then eight distinct parts (with two features each), corresponding to individual joints (Figure 2d). Figure 2b shows an intermediate representation, with something between one and four parts. Four distinct columns are visible, but they do not cleanly segregate the features.

Figure 3 plots the decrease in mean reconstruction error (expressed by the energy $E$)

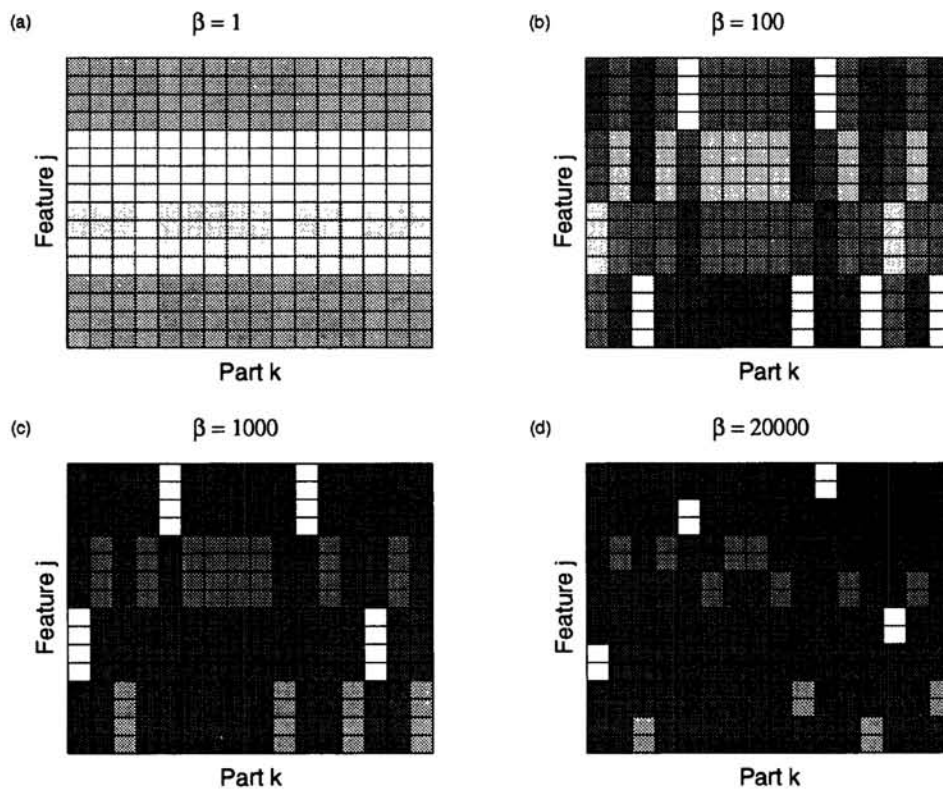

Figure 2: A hierarchy of functional part representations, parameterized by $\beta$.

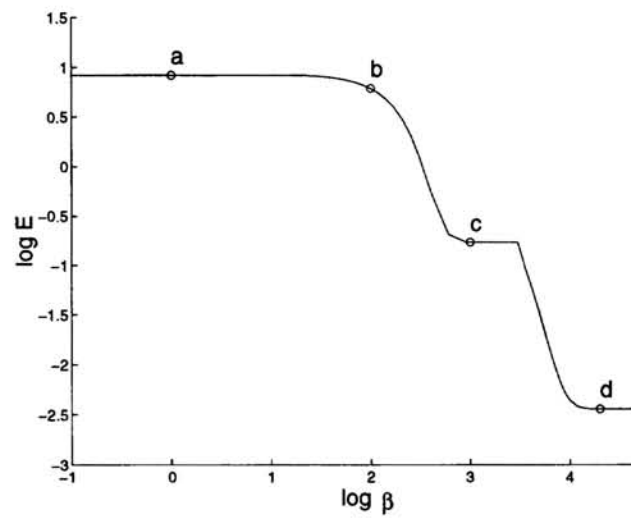

Figure 3: A phase diagram distinguishes true parts (a, c, d) from spurious ones (b).

as $\beta$ increases and more distinct parts emerge. Notice that within the three stable phases corresponding to good part decompositions (Figures 2a, 2c, 2d), $E$ remains practically constant over wide variations in $\beta$. In contrast, $E$ varies rapidly at the boundaries between phases, where spurious part structure appears (Figure 2b). In general, good representations should lie at stable points of this phase diagram, where the error-complexity tradeoff is robust. Thus the actual number of parts in a particular data set, as well as their hierarchical structure, need not be known in advance, but can be inferred from the dynamics of learning.

## 4   LEARNING MULTIPLE CATEGORIES

Until this point, we have assumed that each feature belongs to at most one part over the entire set of training examples, and tried to find the single $K$-part model that best explains the data as a whole. But the notion that a single model must explain the whole data set is quite restrictive. The data may contain several categories of examples, each characterized by a different pattern of feature correlations, and then we would like to learn a *set* of models, each capturing the distinctive part structure of one such category. Again we are motivated by human vision, which easily recognizes many categories of motion defined by high-level patterns of coordinated part movement, such as hand gestures and facial expressions.

If we know which examples belong to which categories, learning multiple models is no harder than learning one, as in the previous section. A separate model can be fit to each category $m$ of training examples, and the weights $p_{jk}^m w_{jk}^m$ are frozen to produce a set of category templates. However, if the category identities are unknown, we face a novel kind of hierarchical learning task. We must simultaneously discover the optimal clustering of examples into categories, as well as the optimal clustering of features into parts *within* each category. We can formalize this hierarchical clustering problem as minimizing a familiar free energy,

$$F = \sum_i \sum_m g^{im}(T^{im} + \frac{1}{\alpha}\log g^{im}),    \tag{7}$$

in which $g^{im}$ specifies the probability of assigning example $i$ to category $m$, and $T^{im}$ is the associated cost. This cost is itself the free energy of the $m$th $K$-part model on the $i$th example,

$$T^{im} = \sum_j \sum_k p_{jk}^m(E_{jk}^{im} + \frac{1}{\beta}\log p_{jk}^m),    \tag{8}$$

in which $p_{jk}^m$ specifies the probability of assigning feature $j$ to part $k$ within category $m$, and $E_{jk}^{im} = (x_j^i - w_{jk}^m y_k^{im})^2$ is the usual reconstruction error from Section 2.

This algorithm was tested on a data set of 256 hand configurations with 20 features each (similar to those in Figure 1), in which each example expresses one of four possible "gestures", i.e. patterns of feature correlation. As Table 1 indicates, the five features on each finger are highly correlated across the entire data set, while variable correlations between the four fingers distinguish the gesture categories. Note that a single model with four parts explains the full variance of the data just as well as the actual four-category generating process. However, most of the data

Table 1: The 20 features are grouped into either 2, 3, or 4 functional parts.

| Examples | No. of parts | Part composition | | | |
|---|---|---|---|---|---|
| 1 - 64 | 2 | 1————10 | 11————————20 | | |
| 65 - 127 | 3 | 1————10 | 11—15 | 16—20 | |
| 128 - 192 | 3 | 1—5 | 6—10 | 11————————20 | |
| 193 - 256 | 4 | 1—5 | 6—10 | 11—15 | 16—20 |

can also be explained by one of several simpler models, making the learner's task a challenging balancing act between accuracy and simplicity.

Figure 4 shows a typical representation learned for this data set. The algorithm was configured with $M = 8$ category models (each with $K = 8$ parts), but only four distinct categories of examples are found after annealing on $\alpha$ (holding $\beta$ constant), and their weights $p_{jk}^m w_{jk}^m$ are depicted in Figure 4a. Each category faithfully captures one of the actual generating categories in Table 1, with the correct number and composition of functional parts. Figure 4b depicts the responsibility $g^{im}$ that each learned category $m$ takes for each feature $i$. Notice the inevitable effect of a bias towards simpler representations. Many examples are misassigned relative to Table 1, when categories with fewer degrees of freedom than their true generating categories can explain them almost as accurately.

## 5 CONCLUSIONS AND FUTURE DIRECTIONS

The notion that many data sets are best explained in terms of functionally independent clusters of correlated features resonates with similar proposals of Foldiak (1990), Saund (1995), Hinton and Zemel (1994), and Dayan and Zemel (1995). Our approach is unique in actually formulating the learning task as a clustering problem and explicitly extracting the functional parts of the data. Factorial learning by clustering features has three principal advantages. First, the free energy cost function for clustering yields a natural complexity scale-space of functional part representations, parameterized by $\beta$. Second, the generalized EM learning algorithm is simple and quick, and maps easily onto a network architecture. Third, by nesting free energies, we can seamlessly compose objective functions for quite complex, hierarchical unsupervised learning problems, such as the multiple category, multiple part mixture problem of Section 4.

The primary limitation of our approach is that when the generative model we assume does not in fact apply to the data, the algorithm may fail to recover any meaningful structure. In ongoing work, we are pursuing a more flexible generative model that allows the underlying causes to compete directly for arbitrary feature-example pairs $ij$, rather than limiting competition only to features $j$, as in Section 2, or only to examples $i$, as in conventional mixture models, or segregating competition for examples and features into hierarchical stages, as in Section 4. Because this introduces many more degrees of freedom, robust learning will require additional constraints, such as temporal continuity of examples or spatial continuity of features.

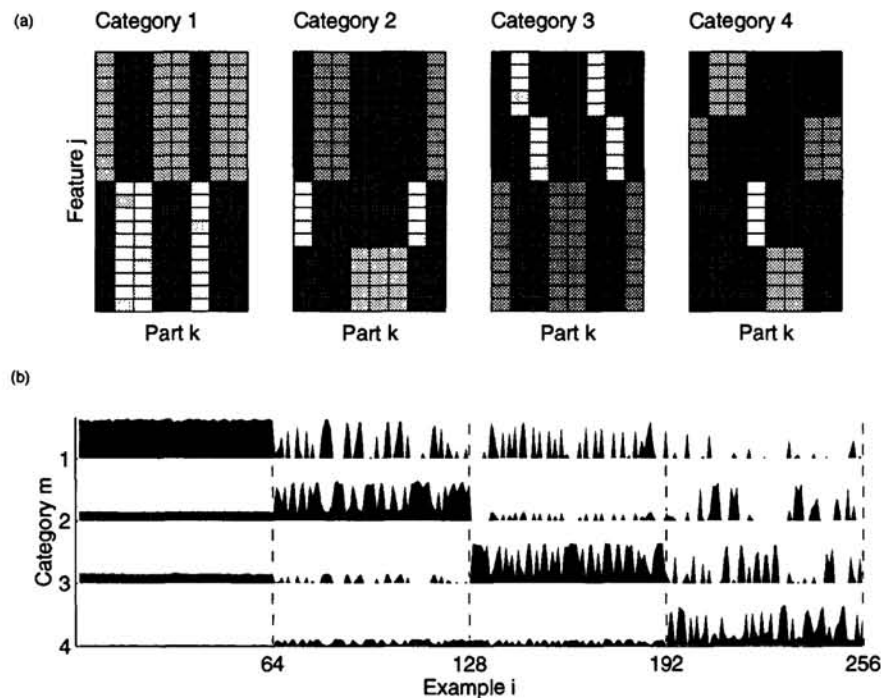

Figure 4: Learning multiple categories, each with a different part structure.

## Acknowledgements

Both authors are Howard Hughes Medical Institute Predoctoral Fellows. We thank Whitman Richards, Yair Weiss, and Stephen Gilbert for helpful discussions.

## References

Dayan, P. and Zemel, R. S. (1995). Competition and multiple cause models. *Neural Computation*, in press.

Foldiak, P. (1990). Forming sparse representations by local anti-hebbian learning. *Biological Cybernetics* **64**, 165-170.

Hinton, G. E. and Zemel, R. S. (1994). Autoencoders, minimum description length and Helmholtz free energy. In J. D. Cowan, G. Tesauro, & J. Alspector (eds.), *Advances in Neural Information Processing Systems 6*. San Mateo, CA: Morgan Kaufmann, 3-10.

Neal, R. M., and Hinton, G. E. (1994). A new view of the EM algorithm that justifies incremental and other variants.

Rose, K., Gurewitz, F., and Fox, G. (1990). Statistical mechanics and phase transitions in clustering. *Physical Review Letters* **65**, 945-948.

Saund, E. (1995). A multiple cause mixture model for unsupervised learning. *Neural Computation* **7**, 51-71.

Tenenbaum, J. (1994). Functional parts. In A. Ram & K. Eiselt (eds.), *Proceedings of the Sixteenth Annual Conference of the Cognitive Science Society*. Hillsdale, NJ: Lawrence Erlbaum, 864-869.